# Bayesian Robustification for Audio Visual Fusion

Javier Movellan *
movellan@cogsci.ucsd.edu
Department of Cognitive Science
University of California, San Diego
La Jolla, CA 92092-0515

Paul Mineiro
pmineiro@cogsci.ucsd.edu
Department of Cognitive Science
University of California, San Diego
La Jolla, CA 92092-0515

## Abstract

We discuss the problem of catastrophic fusion in multimodal recognition systems. This problem arises in systems that need to fuse different channels in non-stationary environments. Practice shows that when recognition modules within each modality are tested in contexts inconsistent with their assumptions, their influence on the fused product tends to increase, with catastrophic results. We explore a principled solution to this problem based upon Bayesian ideas of competitive models and inference robustification: each sensory channel is provided with simple white-noise context models, and the perceptual hypothesis and context are jointly estimated. Consequently, context deviations are interpreted as changes in white noise contamination strength, automatically adjusting the influence of the module. The approach is tested on a fixed lexicon automatic audiovisual speech recognition problem with very good results.

## 1 Introduction

In this paper we address the problem of catastrophic fusion in automatic multimodal recognition systems. We explore a principled solution based on the Bayesian ideas of competitive models and inference robustification (Clark & Yuille, 1990; Box, 1980; O'Hagan, 1994). For concreteness, consider an audiovisual car telephony task which we will simulate in later sections. The task is to recognize spoken phone numbers based on input from a camera and a microphone. We want the recognition system to work on environments with non-stationary statistical properties: at times the video signal (V) may be relatively clean and the audio signal (A) may be contaminated by sources like the radio, the engine, and friction with the road. At other times the A signal may be more reliable than the V signal, e.g., the radio is off, but the talker's mouth is partially occluded. Ideally we want the audio-visual system to combine the A and V sources optimally given the conditions at hand, e.g., give more weight to whichever channel is more reliable at that time. At a minimum we expect that for a wide variety of contexts, the performance after fusion should not be worse than the independent unimodal systems (Bernstein & Benoit, 1996). When component modules can significantly outperform the overall system after fusion, *catastrophic fusion* is said to have occurred.

Fixed vocabulary audiovisual speech recognition (AVSR) systems typically consist of two independent modules, one dedicated to A signals and one to V signals (Bregler, Hild, Manke & Waibel, 1993; Wolff, Prasad, Stork & Hennecke, 1994; Adjondani & Benoit, 1996; Movellan & Chadderdon, 1996). From a Bayesian perspective this modularity reflects an assumption of conditional independence of A and V signals (i.e., the likelihood function factorizes)

$$p(x_a x_v | \omega_i \lambda_a \lambda_v) \propto p(x_a | \omega_i \lambda_a) p(x_v | \omega_i \lambda_v), \tag{1}$$

where $x_a$ and $x_v$ are the audio and video data, $\omega_i$ is a perceptual interpretation of the data (e.g., the word "one") and $\{\lambda_a, \lambda_v\}$ are the audio and video models according to which these probabilities are calculated, e.g., a hidden Markov model, a neural network, or an exemplar model. Training is also typically modularized: the A module is trained to maximize the likelihood of a sample of A signals while the V module is trained on the corresponding sample of V signals. At test time new data are presented to the system and each module typically outputs the log probability of its input given each perceptual alternative. Assuming conditional independence, Bayes' rule calls for an affine combination of modules

$$
\begin{aligned}
\hat{w} &= \underset{\omega_i}{\operatorname{argmax}} \{\log p(\omega_i | x_a x_v \lambda_a \lambda_v)\} \\
&= \underset{\omega_i}{\operatorname{argmax}} \{\log p(x_a | \omega_i \lambda_a) + \log p(x_v | \omega_i \lambda_v) + \log p(\omega_i)\}
\end{aligned}
\tag{2}
$$

where $\hat{w}$ is the interpretation chosen by the system, and $p(w_i)$ is the prior probability of each alternative. This fusion rule is optimal in the sense that it minimizes the expected error: no other fusion rule produces smaller error rates, provided the models $\{\lambda_a, \lambda_v\}$ and the assumption of conditional independence are correct.

Unfortunately a naive application of Bayes' rule to AVSR produces catastrophic fusion. The A and V modules make assumptions about the signals they receive, either explicitly, e.g., a well defined statistical model, or implicitly, e.g., a blackbox trained with a particular data sample. In our notation these assumptions are reflected by the fact that the log-likelihoods are conditional on models: $\{\lambda_a, \lambda_v\}$. The fact that modules make assumptions implies that they will operate correctly only within a restricted *context*, i.e, the collection of situations that meet the assumptions. In practice one typically finds that Bayes' rule assigns more weight to modules operating outside their valid context, the opposite of what is desired.

## 2 Competitive Models and Bayesian Robustification

Clark and Yuille (1990) and Yuille and Bulthoff (1996) analyzed information integration in sensory systems from a Bayesian perspective. Modularity is justified in their view by the need to make assumptions that disambiguate the data available to the perceptual system (Clark & Yuille, 1990, p. 5). However, this produces modules which are valid only within certain contexts. The solution proposed by Clark and Yuille (1990) is the creation of an ensemble of models each of which specializes on a restricted context and automatically checks whether the context is correct. The hope is that by working with such an ensemble of models, robustness under a variety of contexts can be achieved (Clark & Yuille, 1990, p. 13).

Box (1980) investigated the problem of robust statistical inference from a Bayesian perspective. He proposed extending inference models with additional "nuisance" parameters $\sigma$, a process he called Bayesian robustification. The idea is to replace an implicit assumption about the specific value of $\sigma$ with a prior distribution over $\sigma$, representing uncertainty about that parameter.

The approach here combines the ideas of competitive models and robustification. Each of the channels in the multimodal recognition system is provided with extra

parameters that represent non-stationary properties of the environment, what we call a context model. By doing so we effectively work with an infinite ensemble of models each of which compete on-line to explain the data. As we will see later even unsophisticated context models provide superior performance when the environment is non-stationary.

We redefine the estimation problem as simultaneously choosing the most probable A and V context parameters and the most probable perceptual interpretation

$$\hat{w} = \operatorname*{argmax}_{\omega_i} \left\{ \max_{\sigma_a, \sigma_v} p(\omega_i \sigma_a \sigma_v | x_a x_v \lambda_a \lambda_v) \right\} \tag{3}$$

where $\sigma_a$ and $\sigma_v$ are the context parameters for the audio and visual channels and $\omega_i$ are the different perceptual interpretations. One way to think of this joint decision approach is that we let all context models compete and we let only the most probable context models have an influence on the fused percept. Hereafter we refer to this approach as *competitive fusion*.

Assuming conditional independence of the audio and video data and uninformative priors for $(\sigma_a, \sigma_v)$, we have

$$\hat{w} = \operatorname*{argmax}_{\omega_i} \left\{ \log p(\omega_i) + \left[ \max_{\sigma_a} \log p(x_a | \omega_i \sigma_a \lambda_a) \right] + \left[ \max_{\sigma_v} \log p(x_v | \omega_i \sigma_v \lambda_v) \right] \right\}. \tag{4}$$

Thus conditional independence allows a modular implementation of competitive fusion, i.e., the A and V channels do not need to talk to each other until the time to make a joint decision, as follows.

1. For each $\omega_i$ obtain conditional estimates of the context parameters for the audio and video signals:

$$\hat{\sigma}^2_{a|\omega_i} \triangleq \operatorname*{argmax}_{\sigma_a} \left\{ \log p(x_a | \omega_i \sigma_a \lambda_a) \right\}, \tag{5}$$

and

$$\hat{\sigma}^2_{v|\omega_i} \triangleq \operatorname*{argmax}_{\sigma_v} \left\{ \log p(x_v | \omega_i \sigma_v \lambda_v) \right\}. \tag{6}$$

2. Find the best $\omega_i$ using the conditional context estimates.

$$\hat{w} = \operatorname*{argmax}_{\omega_i} \left\{ \log p(\omega_i) + \log p(x_a | \omega_i \hat{\sigma}_{a|\omega_i} \lambda_a) + \log p(x_v | \omega_i \hat{\sigma}_{v|\omega_i} \lambda_v) \right\} \tag{7}$$

## 3  Application to AVSR

Competitive fusion can be easily applied to Hidden Markov Models (HMM), an architecture closely related to stochastic neural networks and arguably the most successful for AVSR. Typical hidden Markov models used in AVSR are defined by

- Markovian state dynamics: $p(q_{t+1}|\underline{q}_t) = p(q_{t+1}|q_t)$, where $q_t$ is the state at time $t$ and $\underline{q}_t = (q_1, \cdots q_t)$,
- Conditionally independent sensor models linking observations to states $f(x_t|q_t)$, typically a mixture of multivariate Gaussian densities

$$f(x_t|q_t) = \sum_i p(m_i|q_t)(2\pi)^{-N/2} |\Sigma|^{-1/2} \exp(d(x_t, q_t, \mu_i, \Sigma)), \tag{8}$$

where $N$ is the dimensionality of the data, $m_i$ is the mixture label, $p(m_i|q_t)$ is the mixture distribution for state $q_t$, $\mu_i$ is the centroid for mixture $m_i$, $\Sigma$ is a covariance matrix, and $d$ is the Mahalanobis norm

$$d(x_t, q_t, \mu_i, \Sigma) = (x_t - \mu_i)'\Sigma^{-1}(x_a - \mu_i). \tag{9}$$

The approach explored here consists on modeling contextual changes as variations on the variance parameters. This corresponds to modeling non-stationary properties of the environments as variations in white noise power within each channel. Competitive fusion calls for on-line maximization of the variance parameters at the same time we optimize with respect to the response alternative.

$$\hat{w} = \underset{\omega_i}{\operatorname{argmax}} \left\{ \log p(\omega_i) + \left[ \max_{\Sigma_a} \log p(x_a|\omega_i \Sigma_a \lambda_a) \right] + \left[ \max_{\Sigma_v} \log p(x_v|\omega_i \Sigma_v \lambda_v) \right] \right\}. \tag{10}$$

The maximization with respect to the variances can be easily integrated into standard HMM packages by simply applying the EM learning algorithm (Dampster, Laird & Rubin, 1977) on the variance parameters at test time. Thus the only difference between the standard approach and competitive fusion is that we retrain the variance parameters of each HMM at test time. In practice this training takes only one or two iterations of the EM algorithm and can be done on-line. We tested this approach on the following AVSR problem.

**Training database** We used Tulips1 (Movellan, 1995) a database consisting of 934 images of 9 male and 3 female undergraduate students from the Cognitive Science Department at the University of California, San Diego. For each of these, two samples were taken for each of the digits "one" through "four". Thus, the total database consists of 96 digit utterances. The specifics of this database are explained in (Movellan, 1995). The database is available at http://cogsci.ucsd.edu.

**Visual processing** We have tried a wide variety of visual processing approaches on this database, including decomposition with local Gaussian templates (Movellan, 1995), PCA-based templates (Gray, Movellan & Sejnowski, 1997), and Gabor energy templates (Movellan & Prayaga, 1996). To date, best performance was achieved with the local Gaussian approach. Each frame of the video track is soft-thresholded and symmetrized along the vertical axis, and a temporal difference frame is obtained by subtracting the previous symmetrized frame from the current symmetrized frame. We calculate the inner-products between the symmetrized images and a set of basis images. Our basis images were 10x15 shifted Gaussian kernels with a standard deviation of 3 pixels. The loadings of the symmetrized image and the differential image are combined to form the final observation frame. Each of these composite frames has 300 dimensions (2x10x15). The process is explained in more detail in Movellan (1995).

**Auditory processing** LPC/cepstral analysis is used for the auditory front-end. First, the auditory signal is passed through a first-order emphasizer to spectrally flatten it. Then the signal is separated into non-overlapping frames at 30 frames per second. This is done so that there are an equal number of visual and auditory feature vectors for each utterance, which are then synchronized with each other. On each frame we perform the standard LPC/cepstral analysis. Each 30 msec auditory frame is characterized by 26 features: 12 cepstral coefficients, 12 delta-cepstrals, 1 log-power, and 1 delta-log-power. Each of the 26 features is encoded with 8-bit accuracy.

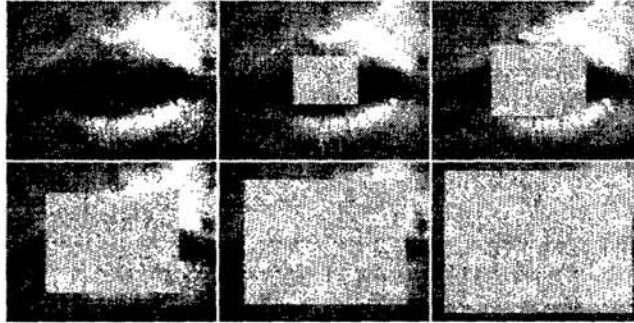

Figure 1: Examples of the different occlusion levels, from left to right: 0%, 10%, 20%, 40%, 60%, 80%. Percentages are in terms of area.

**Recognition Engine**  In previous work (Chadderdon & Movellan, 1995) a wide variety of HMM architectures were tested on this database including architectures that did not assume conditional independence. Optimal performance was found with independent A and V modules using variance matrices of the form $\sigma I$, where $\sigma$ is a scalar and $I$ the identity matrix. The best A models had 5 states and 7 mixtures per state and the best V models had 3 states and 3 mixtures per state. We also determined the optimal weight of A and V modules. Optimal performance is obtained by weighting the output of V times 0.18.

**Factorial Contamination Experiment**  In this experiment we used the previously optimized architecture and compared its performance under 64 different conditions using the standard and the competitive fusion approaches. We used a $2 \times 8 \times 8$ factorial design, the first factor being the fusion rule, and the second and third factors the context in the audio and video channels. To our knowledge this is the first time an AVSR system is tested with a factorial experimental design with both A and V contaminated at various levels. The independent variables were:

1. Fusion rule: Classical, and competitive fusion.
2. Audio Context: Inexistent, clean, or contaminated at one of the following signal to noise ratios: 12 Db, 6 Db, 0 Db, -6 Db, -12 Db and -100 Db. The contamination was done with audio digitally sampled from the interior of a car while running on a busy highway with the doors open and the radio on a talk-show station.
3. Video Context: Inexistent, clean or occluded by a grey level patch. The percentages of visual area occupied by the patch were 10%, 20%, 40%, 60%, 80% and 100% (see Figure 1).

The dependent variable was performance on the digit recognition task evaluated in terms of generalization to new speakers. In all cases training was done with clean signals and testing was done with one of the 64 contexts under study. Since the training sample is small, generalization performance was estimated using a jackknife procedure (Efron, 1982). Models were trained with 11 subjects, leaving a different subject out for generalization testing. The entire procedure was repeated 12 times, each time leaving a different subject out for testing. Statistics of generalization performance are thus based on 96 generalization trials (4 digits × 12 subjects × 2 observations per subject). Standard statistical tests were used to compare the classical and competitive context rules.

The results of this experiment are displayed in Table 1. Note how the experiment replicates the phenomenon of catastrophic fusion. With the classic approach, when one of the channels is contaminated, performance after fusion can be significantly

*Performance with Competitive Fusion*

**Audio**

| Video | None | Clean | 12 Db | 6 Db | 0 Db | -6 Db | -12 Db | -100 Db |
|---|---|---|---|---|---|---|---|---|
| None | — | 95.83 | 95.83 | 90.62 | 80.20 | 67.70 | 42.70 | 19.80 |
| Clean | 84.37 | 97.92 | 97.92 | 94.80 | 90.62 | 89.58 | 81.25 | 82.20 |
| 10% | 73.95 | 93.75 | 93.75 | 94.79 | 87.50 | 80.20 | 71.87 | 64.58 |
| 20% | 62.50 | 96.87 | 96.87 | 94.79 | 89.58 | 80.20 | 62.50 | 41.66 |
| 40% | 37.50 | 93.75 | 89.58 | 87.50 | 83.30 | 70.83 | 43.75 | 30.20 |
| 60% | 34.37 | 93.75 | 91.66 | 88.54 | 82.29 | 65.62 | 42.70 | 26.04 |
| 80% | 27.00 | 95.83 | 90.62 | 86.45 | 79.16 | 64.58 | 46.87 | 25.00 |
| 100% | 25.00 | 93.75 | 92.71 | 84.37 | 78.12 | 63.54 | 44.79 | 26.04 |

*Performance with Classic Fusion*

**Audio**

| Video | None | Clean | 12 Db | 6 Db | 0 Db | -6 Db | -12 Db | -100 Db |
|---|---|---|---|---|---|---|---|---|
| None | — | 95.83 | 94.79 | 89.58 | 79.16 | 65.62 | 40.62 | 20.83 |
| Clean | 86.45 | 98.95 | 96.87 | 95.83 | 93.75 | 87.50 | 79.16 | 70.83 |
| 10% | 73.95 | 93.75 | 93.75 | 93.75 | 89.58 | 79.16 | 70.83 | 52.58 |
| 20% | 54.16 | 89.58 | 84.41 | 84.37 | 84.37 | 75.00 | 51.00 | 43.00 |
| 40% | 29.16 | 81.25 | 78.12 | 78.12 | 67.20 | 52.08 | 38.54 | 34.37 |
| 60% | 32.29 | 77.08 | 77.08 | 72.91 | 62.50 | 47.91 | 37.50 | 29.16 |
| 80% | 29.16 | 70.83 | 72.91 | 68.75 | 54.16 | 44.79 | 33.83 | 28.12 |
| 100% | 25.00 | 61.46 | 61.45 | 58.33 | 51.04 | 42.70 | 38.54 | 29.16 |

Table 1: Average generalization performance with standard and competitive fusion. Boxed cells indicate a statistically significant difference $\alpha = 0.05$ between the two fusion approaches.

worse than performance with the clean channel alone. For example, when the audio is clean, the performance of the audio-only system is 95.83%. When combined with bad video (100% occlusion), this performance drops down to 61.46%, a statistically significant difference, $F(1,11) = 132.0$, $p < 10^{-6}$. Using competitive fusion, the performance of the joint system is 93.75%, which is not significantly different from the performance of the A system only, $F(1,11) = 2.4$, $p = 0.15$. The table shows in boxes the regions for which the classic and competitive fusion approaches were significantly different ($\alpha = 0.05$). Contrary to the classic approach, the competitive approach behaves robustly in all tested conditions.

## 4 Discussion

Catastrophic fusion may occur when the environment is non-stationary forcing modules to operate outside their assumed context. The reason for this problem is that in the absence of a context model, deviations from the expected context are interpreted as information about the different perceptual interpretations instead of information about contextual changes. We explored a principled solution to this problem inspired by the Bayesian ideas of robustification (Box, 1980) and competitive models (Clark & Yuille, 1990). Each module was provided with simple white-noise context models and the most probable context and perceptual hypothesis were jointly estimated. Consequently, context deviations are interpreted as changes in the white noise contamination strength, automatically adjusting the influence of the module. The approach worked very well on a fixed lexicon AVSR problem.

## Footnotes

* To whom correspondence should be addressed.

## References

Adjondani, A. & Benoit, C. (1996). On the Integration of Auditory and Visual Parameters in an HMM-based ASR. In D. G. Stork & M. E. Hennecke (Eds.), *Speechreading by Humans and Machines: Models, Systems, and Applications*, pages 461–471. New York: NATO/Springer-Verlag.

Bernstein, L. & Benoit, C. (1996). For Speech Perception Three Senses are Better

than One. In *Proc. of the 4th Int. Conf. on Spoken Language Processing*, Philadelphia, PA., USA.

Box, G. E. P. (1980). Sampling and Bayes inference in scientific modeling. *J. Roy. Stat. Soc., A., 143*, 383–430.

Bregler, C., Hild, H., Manke, S., & Waibel, A. (1993). Improving Connected Letter Recognition by Lipreading. In *Proc. Int. Conf. on Acoust., Speech, and Signal Processing*, volume 1, pages 557–560, Minneapolis. IEEE.

Bülthoff, H. H. & Yuille, A. L. (1996). A Bayesian framework for the integration of visual modules. In T. Inui & J. L. McClelland (Eds.), *Attention and performance XVI: Information integration in perception and communication*, pages 49–70. Cambridge, MA: MIT Press.

Chadderdon, G. & Movellan, J. (1995). Testing for Channel Independence in Bimodal Speech Recognition. In *Proceedings of 2nd Joint Symposium on Neural Computation*, pages 84–90.

Clark, J. J. & Yuille, A. L. (1990). *Data Fusion for Sensory Information Processing Systems*. Boston: Kluwer Academic Publishers.

Dampster, A. P., Laird, N. M., & Rubin, D. B. (1977). Maximum likelihood from incomplete data via the EM algorithm. *J. Roy. Stat. Soc., 39*, 1–38.

Efron, A. (1982). *The jacknife, the bootstrap and other resampling plans*. Philadelphia, Pennsylvania: SIAM.

Gray, M. S., Movellan, J. R., & Sejnowski, T. (1997). Dynamic features for visual speechreading: A systematic comparison. In Mozer, Jordan, & Petsche (Eds.), *Advances in Neural Information Processing Systems*, volume 9. MIT Press.

Movellan, J. R. (1995). Visual speech recognition with stochastic neural networks. In G. Tesauro, D. Touretzky, & T. Leen (Eds.), *Advances in neural information processing systems*. Cambridge, Massacusetts: MIT Press.

Movellan, J. R. & Chadderdon, G. (1996). Channel Separability in the Audio Visual Integration of Speech: A Bayesian Approach. In D. G. Stork & M. E. Hennecke (Eds.), *Speechreading by Humans and Machines: Models, Systems, and Applications*, pages 473–487. New York: NATO/Springer-Verlag.

Movellan, J. R. & Prayaga, R. S. (1996). Gabor Mosaics: A description of Local Orientation Statistics with Applications to Machine Perception. In G. W. Cottrell (Ed.), *proceedings of the Eight Annual Conference of the Cognitive Science Society*, page 817. Mahwah, New Jersey: LEA.

O'Hagan, A. (1994). *Kendall's Advanced Theory of Statistics: Volume 2B, Bayesian Inference*. volume 2B. Cambridge University Press.

Wolff, G. J., Prasad, K. V., Stork, D. G., & Hennecke, M. E. (1994). Lipreading by Neural Networks: Visual Preprocessing, Learning and Sensory Integration. In J. D. Cowan, G. Tesauro, & J. Alspector (Eds.), *Advances in Neural Information Processing Systems*, volume 6, pages 1027–1034. Morgan Kaufmann.